# Recognizing Activities by Attribute Dynamics

**Weixin Li**     **Nuno Vasconcelos**
Department of Electrical and Computer Engineering
University of California, San Diego
La Jolla, CA 92093, United States
{wel017, nvasconcelos}@ucsd.edu

## Abstract

In this work, we consider the problem of modeling the dynamic structure of human activities in the attributes space. A video sequence is first represented in a semantic feature space, where each feature encodes the probability of occurrence of an activity attribute at a given time. A generative model, denoted the *binary dynamic system* (BDS), is proposed to learn *both* the distribution and dynamics of different activities in this space. The BDS is a non-linear dynamic system, which extends both the binary principal component analysis (PCA) and classical linear dynamic systems (LDS), by combining binary observation variables with a hidden Gauss-Markov state process. In this way, it integrates the representation power of semantic modeling with the ability of dynamic systems to capture the temporal structure of time-varying processes. An algorithm for learning BDS parameters, inspired by a popular LDS learning method from dynamic textures, is proposed. A similarity measure between BDSs, which generalizes the Binet-Cauchy kernel for LDS, is then introduced and used to design activity classifiers. The proposed method is shown to outperform similar classifiers derived from the kernel dynamic system (KDS) and state-of-the-art approaches for dynamics-based or attribute-based action recognition.

## 1 Introduction

Human activity understanding has been a research topic of substantial interest in computer vision [1]. Inspired by the success of the popular *bag-of-features* (BoF) representation on image classification problems, it is frequently based on the characterization of video as a collection of orderless spatiotemporal features [2, 3]. Recently, there have been attempts to extend this representation along two dimensions that we explore in this work. The first is to introduce richer models for the temporal structure, also known as *dynamics*, of human actions [4, 5, 6, 7]. This aims to exploit the fact that actions are usually defined as sequences of poses, gestures, or other events over time. While desirable, modeling action dynamics can be a complex proposition, and this can sometimes compromise the robustness of recognition algorithms, or sacrifice their generality, *e.g.*, it is not uncommon for dynamic models to require features specific to certain datasets or action classes [5, 6], or non-trivial forms of pre-processing, such as tracking [8], manual annotation [7], *etc*. The second dimension, again inspired by recent developments in image classification [9, 10], is to represent actions in terms of intermediate-level *semantic concepts*, or *attributes* [11, 12]. This introduces a layer of abstraction that improves the generalization of the representation, enables modeling of contextual relationships [13], and simplifies knowledge transfer across activity classes [11].

In this work, we propose a representation that combines the benefits of these two types of extensions. This consists of *modeling the dynamics of human activities in the attributes space*. The idea is to exploit the fact that an activity is usually defined as a sequence of *semantic* events. For example, the activity "storing an object in a box" is defined as the *sequence* of the action attributes "remove (hand from box)", "grab (object)", "insert (hand in box)", and "drop (object)". The representation of

the action as a sequence of these attributes makes the characterization of the "storing object in box" activity *more robust* (to confounding factors such as diversity of grabbing styles, hand motion speeds, or camera motions) than dynamic representations based on low-level features. It is also *more discriminant* than semantic representations that ignore dynamics, *i.e.*, that simply record the occurrence (or frequency) of the action attributes "remove", "grab", "insert", and "drop". In the absence of information about the *sequence* in which these attributes occur, the "store object in box" activity cannot be distinguished from the "retrieve object from box" activity, defined as the sequence "insert (hand in box)", "grab (object)", "remove (hand from box)", and "drop (object)". In summary, the modeling of attribute dynamics is 1) more robust and flexible than the modeling of visual (low-level) dynamics, and 2) more discriminant than the modeling of attribute frequencies.

In this work, we address the problem of modeling attribute dynamics for activities. As is usual in semantics-based recognition [11], we start by representing video in a semantic feature space, where each feature encodes the probability of occurrence of an action attribute in the video, at a given time. We then propose a generative model, denoted the *binary dynamic system* (BDS), to learn *both* the distribution and dynamics of different activities in this space. The BDS is a non-linear dynamic system, which combines binary observation variables with a hidden Gauss-Markov state process. It can be interpreted as either 1) a generalization of *binary principal component analysis* (binary PCA) [14], which accounts for data dynamics, or 2) an extension of the classical *linear dynamic system* (LDS), which operates on a binary observation space. For activity recognition, the BDS has the appeal of accounting for the two distinguishing properties of the semantic activity representation: 1) that semantic vectors define probability distributions over a space of binary attributes; and 2) that these distributions evolve according to smooth trajectories that reflect the dynamics of the underlying activity. Its advantages over previous representations are illustrated by the introduction of BDS-based activity classifiers. For this, we start by proposing an efficient BDS learning algorithm, which combines binary PCA and a least squares problem, inspired by the learning procedure in *dynamic textures* [15]. We then derive a similarity measure between BDSs, which generalizes the Binet-Cauchy kernel from the LDS literature [16]. This is finally used to design activity classifiers, which are shown to outperform similar classifiers derived from the kernel dynamic systems (KDS) [6], and state-of-the-art approaches for dynamics-based [4] and attribute-based [11] action recognition.

## 2 Prior Work

One of the most popular representations for activity recognition is the BoF, which reduces video to an collection of orderless spatiotemporal descriptors [2, 3]. While robust, the BoF ignores the temporal structure of activities, and has limited power for fine-grained activity discrimination. A number of approaches have been proposed to characterize this structure. One possibility is to represent actions in terms of limb or torso motions, spatiotemporal shape models, or motion templates [17, 18]. Since they require detection, segmentation, tracking, or 3D structure recovery of body parts, these representations can be fragile. A robust alternative is to model the temporal structure of the BoF. This can be achieved with generalizations of popular still image recognition methods. For example, Laptev *et al.* extend pyramid matching to video, using a 3D binning scheme that roughly characterizes the spatio-temporal structure of video [3]. Niebles *et al.* employ a latent SVM that augments the BoF with temporal context, which they show to be critical for understanding realistic motion [4]. All these approaches have relatively coarse modeling of dynamics. More elaborate models are usually based on generative representations. For example, Laxton *et al.* model a combination of object contexts and action sequences with a dynamic Bayesian network [5], while Gaidon *et al.* reduce each activity to three atomic actions and model their temporal distributions [7]. These methods rely on action-class specific features and require detailed manual supervision. Alternatively, several researchers have proposed to model BoF dynamics with LDSs. For example, Kellokumpu *et al.* combine dynamic textures [15] and local binary patterns [19], Li *et al.* perform a discriminant canonical correlation analysis on the space of action dynamics [8], and Chaudhry *et al.* map frame-wise motion histograms to a reproducing kernel Hilbert space (RKHS), where they learn a KDS [6].

Recent research in image recognition has shown that various limitations of the BoF can be overcome with representations of higher semantic level [10]. The features that underly these representations are confidence scores for the appearance of pre-defined visual concepts in images. These concepts can be object attributes [9], object classes [20, 21], contextual classes [13], or generic visual concepts [22]. Lately, semantic attributes have also been used for action recognition [11], demonstrating the benefits of a mid-level semantic characterization for the analysis of complex human activities.

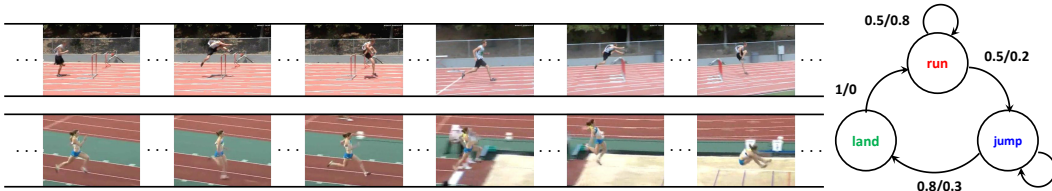

**Figure 1:** Left: key frames of activities "hurdle race" (top) and "long jump" (bottom); Right: attribute transition probabilities of the two activities ("hurdle race" / "long jump") for attributes "run", "jump", and "land".

The work also suggests that, for action categorization, supervised attribute learning is far more useful than unsupervised learning, resembling a similar observation from image recognition [20]. However, all of these representations are BoF-like, in the sense that they represent actions as orderless feature collections, reducing an *entire* video sequence to an attribute vector. For this reason, we denote them *holistic attribute representations.*

The temporal evolution of semantic concepts, throughout a video sequence, has not yet been exploited as a cue for action understanding. There has, however, been some progress towards this type of modeling in the text analysis literature, where temporal extensions of latent Dirichlet allocation (LDA) have been proposed. Two representatives are the dynamic topic model (DTM) [23] and the topic over time (TOT) model [24]. Although modeling topic dynamics, these models are not necessarily applicable to semantic action recognition. First, like the underlying LDA, they are unsupervised models, and thus likely to underperform in recognition tasks [11, 10]. Second, the joint goal of topic discovery and modeling topic dynamics requires a complex graphical model. This is at odds with tractability, which is usually achieved by sacrificing the expressiveness of the temporal model component.

## 3 Modeling the Dynamics of Activity Attributes

In this section, we introduce a new model, the binary dynamic system, for joint representation of the distribution and dynamics of activities in action attribute space.

### 3.1 Semantic Representation

Semantic representations characterize video as a collection of descriptors with *explicit* semantics [10, 11]. They are obtained by defining a set of *semantic concepts* (or *attributes*, *scene classes*, *etc*), and learning a classifier to detect each of those concepts. Given a video $v \in \mathcal{X}$ to analyze, each classifier produces a confidence score for the presence of the associated concept. The ensemble of classifiers maps the video to a *semantic space* $\mathcal{S}$, according to $\boldsymbol{\pi} : \mathcal{X} \to \mathcal{S} = [0,1]^K, \boldsymbol{\pi}(v) = (\pi_1(v), \cdots, \pi_K(v))^T$, where $\pi_i(v)$ is the confidence score for the presence of the $i$-th concept. In this work, the classification score is the *posterior probability* of a concept $c$ given video $v$, *i.e.*, $\pi_c(v) = p(c|v)$ under a certain video representation, *e.g.*, the popular BoF histogram of spatio-temporal descriptors. As the video sequence $v$ progresses with time $t$, the semantic encoding defines a trajectory $\{\boldsymbol{\pi}_t(v)\} \subset \mathcal{S}$. The benefits of semantic representations for recognition, namely a higher level of abstraction (which leads to better generalization than appearance-based representations), substantial robustness to the performance of the visual classifiers $\pi_i(v)$, and intrinsic ability to account for contextual relationships between concepts, have been previously documented in the literature [13]. No attention has, however, been devoted to modeling the *dynamics* of semantic encodings of video. Figure 1 motivates the importance of such modeling for action recognition, by considering two activity categories ("long jump" and "hurdle race"), which involve the same attributes, with roughy the same probabilities, but span very different trajectories in $\mathcal{S}$. Modeling these dynamics can substantially enhance the ability of a classifier to discriminate between complex activities.

### 3.2 Binary PCA

The proposed representation is a generalization of binary PCA [14], a dimensionality reduction technique for binary data, belonging to the generalized exponential family PCA [25]. It fits a linear model to binary observations, by embedding the natural parameters of Bernoulli distributions in a low-dimensional subspace. Let $Y$ denote a $K \times \tau$ binary matrix ($Y_{kt} \in \{0, 1\}$, *e.g.*, the indicator of

occurrence of attribute $k$ at time $t$) where each column is a vector of $K$ binary observations sampled from a multivariate Bernoulli distribution

$$Y_{kt} \sim B(y_{kt}; \pi_{kt}) = \pi_{kt}^{y_{kt}}(1 - \pi_{kt})^{1-y_{kt}} = \sigma(\theta_{kt})^{y_{kt}}\sigma(-\theta_{kt})^{1-y_{kt}}, \; y_{kt} \in \{0, 1\}. \quad (1)$$

The log-odds $\theta = \log(\frac{\pi}{1-\pi})$ is the natural parameter of the Bernoulli distribution, and $\sigma(\theta) = (1 + e^{-\theta})^{-1}$ is the logistic function. Binary PCA finds a $L$-dimensional ($L \ll K$) embedding of the natural parameters, by maximizing the log-likelihood of the binary matrix $Y$

$$\mathcal{L} = \log P(Y; \Theta) = \sum_{k,t} \left[ Y_{kt} \log \sigma(\Theta_{kt}) + (1 - Y_{kt}) \log \sigma(-\Theta_{kt}) \right] \quad (2)$$

under the constraint

$$\Theta = CX + \boldsymbol{u}\mathbf{1}^T, \quad (3)$$

where $C \in \mathbb{R}^{K \times L}$, $X \in \mathbb{R}^{L \times \tau}$, $\boldsymbol{u} \in \mathbb{R}^K$ and $\mathbf{1} \in \mathbb{R}^\tau$ is the vector of all ones. Each column of $C$ is a basis vector of a latent subspace and the $t$-th column of $X$ contains the coordinates of the $t$-th binary vector in this basis (up to a translation by $\boldsymbol{u}$).

Binary PCA is not directly applicable to attribute-based recognition, where the goal is to fit the vectors of confidence scores $\{\boldsymbol{\pi}_t\}$ produced by a set of $K$ attribute classifiers (and not a sample of binary attribute vectors *per se*). To overcome this problem, we maximize the *expected log-likelihood* of the data $Y$ (which is the lower bound to the log expected likelihood of the data $Y$, by Jensen's inequality). Since $\mathbb{E}[\boldsymbol{y}_t] = \boldsymbol{\pi}_t$, it follows from (2) that

$$\mathbb{E}_Y[\mathcal{L}] = \sum_{k,t} \left[ \pi_{kt} \log \sigma(\Theta_{kt}) + (1 - \pi_{kt}) \log \sigma(-\Theta_{kt}) \right]. \quad (4)$$

The proposed extension of binary PCA consists of maximizing this expected log-likelihood under the constraint of (3). It can be shown that, in the absence of the constraint, the maximum occurs when $\sigma(\Theta_{kt}) = \pi_{kt}, \forall k, t$. As in PCA, (3) forces $\sigma(\Theta_{kt})$ to lie on a subspace of $\mathcal{S}$, *i.e.*,

$$\sigma(\Theta_{kt}) = \hat{\pi}_{kt} \approx \pi_{kt}. \quad (5)$$

The difference between the expected log-likelihood of the true scores $\{\boldsymbol{\pi}_t\}$ and the binary PCA scores $\{\sigma(\boldsymbol{\theta}_t) = \sigma(C\boldsymbol{x}_t + \boldsymbol{u})\}$ $(\sigma(\boldsymbol{\theta}) \equiv [\sigma(\theta_1), \cdots, \sigma(\theta_K)]^T)$ is

$$\mathbb{E}[\Delta\mathcal{L}(\{\boldsymbol{\pi}_t\}; \{\sigma(\boldsymbol{\theta}_t)\})] \quad = \quad \mathbb{E}_Y\left[\log(P(Y; \{\boldsymbol{\pi}_t\}))\right] - \mathbb{E}_Y\left[\log(P(Y; \{\sigma(\boldsymbol{\theta}_t)\}))\right] \quad (6)$$

$$= \quad \sum_{k,t} \left[ \pi_{kt} \log \frac{\pi_{kt}}{\sigma(\Theta_{kt})} + (1 - \pi_{kt}) \log \frac{1 - \pi_{kt}}{\sigma(-\Theta_{kt})} \right] \quad (7)$$

$$= \quad \sum_t \mathrm{KL}[B(\boldsymbol{y}; \boldsymbol{\pi}_t) || B(\boldsymbol{y}; \sigma(\boldsymbol{\theta}_t))], \quad (8)$$

where $\mathrm{KL}(B(\boldsymbol{y}; \boldsymbol{\pi}) || B(\boldsymbol{y}; \boldsymbol{\pi}'))$ is the Kullback-Leibler (KL) divergence between two multivariate Bernoulli distributions of parameters $\boldsymbol{\pi}$ and $\boldsymbol{\pi}'$. By maximizing the expected log-likelihood (4), the optimal projection $\{\boldsymbol{\theta}_t^*\}$ of the attribute score vectors $\{\boldsymbol{\pi}_t\}$ on the subspace of (3) also minimizes the KL divergence of (8). Hence, for the optimal natural parameters $\{\boldsymbol{\theta}_t^*\}$, the approximation of (5) is the best in the sense of KL divergence, the natural similarity measure between probability distributions.

### 3.3 Binary Dynamic Systems

A discrete time linear dynamic system (LDS) is defined by

$$\begin{cases} \boldsymbol{x}_{t+1} &= A\boldsymbol{x}_t + \boldsymbol{v}_t \\ \boldsymbol{y}_t &= C\boldsymbol{x}_t + \boldsymbol{w}_t + \boldsymbol{u} \end{cases}, \quad (9)$$

where $\boldsymbol{x}_t \in \mathbb{R}^L$ and $\boldsymbol{y}_t \in \mathbb{R}^K$ (of mean $\boldsymbol{u}$) are the hidden *state* and *observation* variable at time $t$, respectively; $A \in \mathbb{R}^{L \times L}$ is the state transition matrix that encodes the underlying dynamics; $C \in \mathbb{R}^{K \times L}$ the observation matrix that linearly maps the state to the observation space; and $\boldsymbol{x}_1 = \boldsymbol{\mu}_0 + \boldsymbol{v}_0 \sim \mathcal{N}(\boldsymbol{\mu}_0, S_0)$ an initial condition. Both state and observations are subject to additive Gaussian noise processes $\boldsymbol{v}_t \sim \mathcal{N}(\mathbf{0}, Q)$ and $\boldsymbol{w}_t \sim \mathcal{N}(\mathbf{0}, R)$. Since the noise is Gaussian and $L < K$, the matrix $C$ can be interpreted as a PCA basis for the observation space ($L$ eigenvectors of the observation covariance). The state vector $\boldsymbol{x}_t$ then encodes the trajectory of the PCA coefficients (projection on this basis) of the observed data over time. This interpretation is, in fact, at the core of the popular *dynamic texture* (DT) [15] representation for video. While the LDS parameters

---

**Algorithm 1:** Learning a binary dynamic system

---

**Input** : a sequence of attribute score vectors $\{\boldsymbol{\pi}_t\}_{t=1}^{\tau}$, state space dimension $n$.

   Binary PCA: $\{C, X, \boldsymbol{u}\} = $ B-PCA($\{\boldsymbol{\pi}_t\}_{t=1}^{\tau}, n$) using the method of [14].
   Estimate state parameters ($X_{t_1}^{t_2} \equiv \left[\boldsymbol{x}_{t_1}, \cdots, \boldsymbol{x}_{t_2}\right]$):
     $A = X_2^{\tau}(X_1^{\tau-1})^{\dagger}; \quad V = (X)_2^{\tau} - A(X)_1^{\tau-1}; \quad Q = \frac{1}{\tau-1}V(V)^T;$
     $\boldsymbol{\mu}_0 = \frac{1}{\tau}\sum_{t=1}^{\tau}\boldsymbol{x}_t; \quad S_0 = \frac{1}{\tau-1}\sum_{t=1}^{\tau}(\boldsymbol{x}_t - \boldsymbol{\mu}_0)(\boldsymbol{x}_t - \boldsymbol{\mu}_0)^T.$

**Output**: $\{A, C, Q, \boldsymbol{u}, \boldsymbol{\mu}_0, S_0\}$

---

can be learned by maximum likelihood, using an expectation-maximization (EM) algorithm [26], the DT decouples the learning of observation and state variables. Observation parameters are first learned by PCA, and state parameters are then learned with a least squares procedure. This simple approximate learning algorithm tends to perform very well, and is widely used in computer vision.

The proposed *binary dynamic system* (BDS) is defined as

$$
\begin{cases}
\boldsymbol{x}_{t+1} &= A\boldsymbol{x}_t + \boldsymbol{v}_t \\
\boldsymbol{y}_t &\sim B(\boldsymbol{y}; \sigma(C\boldsymbol{x}_t + \boldsymbol{u}))
\end{cases}, \tag{10}
$$

where $\boldsymbol{x}_t \in \mathbb{R}^L$ and $\boldsymbol{u} \in \mathbb{R}^K$ are the hidden state variable and observation bias, respectively; $A \in \mathbb{R}^{L \times L}$ is the state transition matrix; and $C \in \mathbb{R}^{K \times L}$ the observation matrix. The initial condition is given by $\boldsymbol{x}_1 = \boldsymbol{\mu}_0 + \boldsymbol{v}_0 \sim \mathcal{N}(\boldsymbol{\mu}_0, S_0)$; and the state noise process is $\boldsymbol{v}_t \sim \mathcal{N}(\boldsymbol{0}, Q)$. Like the LDS of (9), the BDS can be interpreted as combining a (now binary) PCA observation component with a Gauss-Markov process for the state sequence. As in binary PCA, for attribute-based recognition the binary observations $\boldsymbol{y}_t$ are replaced by the attribute scores $\boldsymbol{\pi}_t$, their log-likelihood under (10) by the expected log-likelihood, and the optimal solution minimizes the approximation of (5) for the most natural definition of similarity (KL divergence) between probability distributions. This is conceptually equivalent to the behavior of the canonical LDS of (9), which determines the subspace that best approximates the observations in the *Euclidean* sense, the natural similarity measure for Gaussian data. Note that other extensions of the LDS, *e.g.*, kernel dynamic systems (KDS) that rely on a non-linear kernel PCA (KPCA) [27] of the observation space but still assume an Euclidean measure (Gaussian noise) [28, 6], do not share this property. We will see, in the experimental section, that the BDS is a better model of attribute dynamics.

### 3.4 Learning

Since the Gaussian state distribution of an LDS is a conjugate prior for the (Gaussian) conditional-distribution of its observations given the state, maximum-likelihood estimates of LDS parameters are tractable. The LDS parameters $\boldsymbol{\Omega}_{LDS} = \{A, C, Q, R, \boldsymbol{\mu}_0, S_0, \boldsymbol{u}\}$ of (9) can thus be estimated with an EM algorithm [26]. For the BDS, where the state is Gaussian but the observations are not, the expectation step is intractable. Hence, approximate inference is required to learn the parameters $\boldsymbol{\Omega}_{BDS} = \{A, C, Q, \boldsymbol{\mu}_0, S_0, \boldsymbol{u}\}$ of (10). In this work, we resort to the approximate DT learning procedure, where observation and state components are learned separately [15]. The binary PCA basis is learned first, by maximizing the expected log-likelihood of (4) subject to the constraint of (3). Since the Bernoulli distribution is a member of exponential family, (4) is concave in $\Theta$, but not in $C, X$ and $\boldsymbol{u}$ jointly. We rely on a procedure introduced by [14], which iterates between the optimization with respect to one of the variables $C, X$ and $\boldsymbol{u}$, with the remaining two held constant. Each iteration is a convex sub-problem that can be solved efficiently with a fixed-point auxiliary function (see [14] for details). Once the latent embedding $C^*, X^*$ and $\boldsymbol{u}^*$ of the attribute sequence in the optimal subspace is recovered, the remaining parameters are estimated by solving a least-squares problem for $A$ and $Q$, and using standard maximum likelihood estimates for the Gaussian parameters of the initial condition ($\boldsymbol{\mu}_0$ and $S_0$) [15]. The procedure is summarized in Algorithm 1.

## 4 Measuring Distances between BDSs

The design of classifiers that account for attribute dynamics requires the ability to quantify similarity between BDSs. In this section, we derive the BDS counterpart to the popular Binet-Cauchy kernel (BCK) for the LDS, which evaluates the similarity of the output sequences of two LDSs. Given

LDSs $\mathbf{\Omega}_a$ and $\mathbf{\Omega}_b$ driven by identical noise processes $\boldsymbol{v}_t$ and $\boldsymbol{w}_t$ with observation sequences $\boldsymbol{y}^{(a)}$ and $\boldsymbol{y}^{(b)}$, [16] propose a family of BCKs

$$K_{BC}(\mathbf{\Omega}_a, \mathbf{\Omega}_b) = \mathbb{E}_{\boldsymbol{v},\boldsymbol{w}}\left[\sum_{t=0}^{\infty} e^{-\lambda t}(\boldsymbol{y}_t^{(a)})^T W \boldsymbol{y}_t^{(b)}\right], \qquad (11)$$

where $W$ is a semi-definite positive weight matrix and $\lambda \geqslant 0$ a temporal discounting factor. To extend (11) to BDSs $\mathbf{\Omega}_a$ and $\mathbf{\Omega}_b$, we note that $(\boldsymbol{y}_t^{(a)})^T W \boldsymbol{y}_t^{(b)}$ is the inner product of an Euclidean output space of metric $d^2(\boldsymbol{y}_t^{(a)}, \boldsymbol{y}_t^{(b)}) = (\boldsymbol{y}_t^{(a)} - \boldsymbol{y}_t^{(b)})^T W(\boldsymbol{y}_t^{(a)} - \boldsymbol{y}_t^{(b)})$. For BDSs, whose observations $\boldsymbol{y}_t$ are Bernouli distributed with parameters $\{\sigma(\boldsymbol{\theta}_t^{(a)})\}$, for $\mathbf{\Omega}_a$, and $\{\sigma(\boldsymbol{\theta}_t^{(b)})\}$, for $\mathbf{\Omega}_b$, this distance measure is naturally replaced by the KL divergence between Bernoulli distributions

$$
\begin{aligned}
D_{BC}(\mathbf{\Omega}_a, \mathbf{\Omega}_b) &= \mathbb{E}_{\boldsymbol{v}}\left[\sum_{t=0}^{\infty} e^{-\lambda t}\left(\mathrm{KL}(B(\sigma(\boldsymbol{\theta}_t^{(a)}))||B(\sigma(\boldsymbol{\theta}_t^{(b)}))) + \mathrm{KL}(B(\sigma(\boldsymbol{\theta}_t^{(b)}))||B(\sigma(\boldsymbol{\theta}_t^{(a)})))\right)\right] \\
&= \mathbb{E}_{\boldsymbol{v}}\left[\sum_{t=0}^{\infty} e^{-\lambda t}\left(\sigma(\boldsymbol{\theta}_t^{(a)}) - \sigma(\boldsymbol{\theta}_t^{(b)})\right)^T \left(\boldsymbol{\theta}_t^{(a)} - \boldsymbol{\theta}_t^{(b)}\right)\right],
\end{aligned}
\qquad (12)
$$

where $\boldsymbol{\theta}_t = C\boldsymbol{x}_t + \boldsymbol{u}$. The distance term at time $t$ can be rewritten as

$$(\sigma(\boldsymbol{\theta}_t^{(a)}) - \sigma(\boldsymbol{\theta}_t^{(b)}))^T(\boldsymbol{\theta}_t^{(a)} - \boldsymbol{\theta}_t^{(b)}) = (\boldsymbol{\theta}_t^{(a)} - \boldsymbol{\theta}_t^{(b)})^T \hat{W}_t(\boldsymbol{\theta}_t^{(a)} - \boldsymbol{\theta}_t^{(b)}), \qquad (13)$$

with $\hat{W}_t$ a diagonal matrix whose $k$-th diagonal element is $\hat{W}_{t,k} = (\sigma(\Theta_{t,k}^{(a)}) - \sigma(\Theta_{t,k}^{(b)}))/(\Theta_{t,k}^{(a)} - \Theta_{t,k}^{(b)}) = \sigma'(\hat{\Theta}_{t,k}^{(a,b)})$ (where, by the mean value theorem, $\hat{\Theta}_{t,k}^{(a,b)}$ is some real value between $\hat{\Theta}_{t,k}^{(a)}$ and $\hat{\Theta}_{t,k}^{(b)}$). This reduces (13) to a form similar to (11), although with a time varying weight matrix $W_t$. It is unclear whether (12) can be computed in closed-form. We currently rely on the approximation $D_{BC}(\mathbf{\Omega}_a, \mathbf{\Omega}_b) \approx \sum_{t=0}^{\infty} e^{-\lambda t}(\sigma(\bar{\boldsymbol{\theta}}_t^{(a)}) - \sigma(\bar{\boldsymbol{\theta}}_t^{(b)}))^T(\bar{\boldsymbol{\theta}}_t^{(a)} - \bar{\boldsymbol{\theta}}_t^{(b)})$, where $\bar{\boldsymbol{\theta}}$ is the mean of $\boldsymbol{\theta}$.

## 5 Experiments

Several experiments were conducted to evaluate the BDS as a model of activity attribute dynamics. In all cases, the BoF was used as low-level video representation, interest points were detected with [2], and HoG/HoF descriptors [3] computed at their locations. A codebook of 3000 visual words was learned via $k$-means, from the entire training set, and a binary SVM with histogram intersection kernel (HIK) and probability outputs [29] trained to detect each attribute using the attribute definition same as [11]. The probability for attribute $k$ at time $t$ was used as attribute score $\pi_{tk}$, which was computed over a window of 20 frames, sliding across a video.

### 5.1 Weizmann Activities

To obtain some intuition on the performance of different algorithms considered, we first used complex activity sequences synthesized from the Weizmann dataset [17]. This contains 10 *atomic* action classes (*e.g.*, skipping, walking) annotated with respect to 30 lower-level attributes (*e.g.*, "one-arm-motion"), and performed by 9 people. We created *activity sequences* by concatenating Weizmann actions. A sequence of *degree* $n$ ($n = 4, 5, 6$) is composed of $n$ atomic actions, performed by the same person. The row of images at the top of Figure 2 presents an example of an activity sequence of degree 5. The images shown at the top of the figure are keyframes from the atomic actions ("walk", "pjump", "wave1", "wave2", "wave2") that compose this activity sequence. The black curve (labeled "Sem. Seq") in the plot at the bottom of the figure shows the score of the "two-arms-motion" attribute, as a function of time. 40 activity categories were defined per degree $n$ (total of 120 activity categories) and a dataset was assembled per category, containing one activity sequence per person (9 people, 1080 sequences in total). Overall, the activity sequences differ in the number, category, and temporal order of atomic actions. Since the attribute ground truth is available for all atomic actions in this dataset, it is possible to train clean attribute models. Hence, all performance variations can be attributed to the quality of the attribute-based inference of different approaches.

We started by comparing the binary PCA representation that underlies the BDS to the PCA and KPCA decompositions of the LDS and KDS. In all cases we projected a set of attribute score vectors $\{\boldsymbol{\pi}_t\}$ into the low-dimensional PCA subspace, computed the reconstructed score vectors $\{\hat{\boldsymbol{\pi}}_t\}$, and the KL divergence $\mathrm{KL}(B(\boldsymbol{y}, \boldsymbol{\pi}_t)||B(\boldsymbol{y}, \hat{\boldsymbol{\pi}}_t))$, as reported in Figure 3. The kernel used for KPCA was

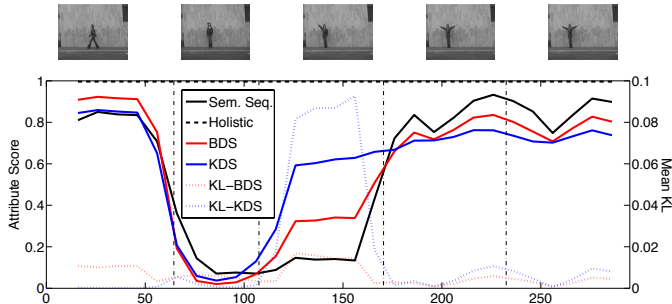

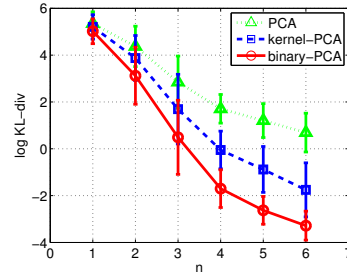

**Figure 2:** Top: key frames from the activity sequence class "walk-pjump-wave1-wave2-wave2". Bottom: score of "two-arms-motion" attribute for video of this activity. True scores in black, and scores sampled from the BDS (red) and KDS (blue). Also shown is the KL-divergence between sampled and original scores, for both models.

**Figure 3:** Log KL-divergence between original and reconstructed attribute scores, *v.s.* number of PCA components $n$, on Weizmann activities for PCA, kernel PCA, and binary PCA.

**Table 1:** Classification Accuracy on Weizmann Activities and Olympic Sports Datasets

| Dataset | BoF | Holistic Attri. | DTM | TOT | KDS | BDS |
|---|---|---|---|---|---|---|
| **Weizmann Activities** | 57.8% | 72.6% | 84.6% | 88.2% | 90.2% | 94.8% |
| **Olympic Sports** | 56.8% | 63.5% | 47.1% | 53.3% | 62.3% | 65.7% |

the logit kernel $K(\boldsymbol{\pi}_1, \boldsymbol{\pi}_2) = \sigma^{-1}(\boldsymbol{\pi}_1)^T \sigma^{-1}(\boldsymbol{\pi}_2)$, where $\sigma^{-1}(\cdot)$ is the element-wise logit function. Figure 3 shows the average log-KL divergence, over the entire dataset, as a function of the number of PCA components used in the reconstruction. Binary PCA outperformed both PCA and KPCA. The improvements over KPCA are particularly interesting since the latter uses the logistic transformation that distinguishes binary PCA from PCA. This is explained by the Euclidean similarity measure that underlies the assumption of Gaussian noise in KPCA, as discussed in Section 3.3. To gain some more insight on the different models, a KDS and a BDS were learned from the 30 dimensional attribute score vectors of the activity sequence in Figure 2. A new set of attribute score vectors were then sampled from each model. The evolution of the scores sampled for the "two-arms-motion" attribute are shown in the figure (in red/blue for BDS/KDS). Note how the scores sampled from the BDS approximate the original attribute scores better than those sampled from the KDS, which is confirmed by the KL-divergences between the original attribute scores and those sampled from the two models (also shown in the figure).

We next evaluated the benefits of different dynamics representations for activity recognition. Recognition rates were obtained with a 9-fold leave-one-out-cross-validation (LOOCV), where, per trial, the activities of one subject were used as test set and those of the remaining 8 as training set. We compared the performance of classifiers based on the KDS and BDS with a BoF classifier, a *holistic attribute classifier* that ignores attribute dynamics (using a single attribute score vector computed from the entire video sequence) and the dynamic topic models DTM [23] and TOT [24] from the text literature. For the latter, the topics were equated to the activity attributes and learned with supervision (using the SVMs discussed above). Unsupervised versions of the topic models had worse performance and are omitted. Classification was performed with Bayes rule for topic models, and a nearest-neighbor classifier for the remaining methods. For BDS, distances were measured with (12), while for the KDS we tried the Binet-Cauchy, $\mathcal{X}^2$, intersection and logit kernels, and reported the best results. $\mathcal{X}^2$ distance was used for the BoF and holistic attribute classifiers. The classification accuracy of all classifiers is shown in Table 1. BDS and KDS had the best performance, followed by the dynamic topic models, and the dynamics insensitive methods (BoF and holistic). Note that the difference between the holistic classifier and the best dynamic model is of approximately $22\%$. This shows that while attributes are important ($14.8\%$ improvement over BoF) they are not the whole story. Problems involving *fine-grained* activity classification, *i.e.*, discrimination between activities composed of similar actions executed in different sequence, requires modeling of attribute dynamics. Among dynamic models, the BDS outperformed the KDS, and topic models DTM and TOT.

## 5.2 Olympic Sports

The second set of experiments was performed on the Olympic Sports dataset [4]. This contains YouTuBe videos of 16 sport activities, with a total of 783 sequences. Some activities are sequences

**Table 2:** Fine-grained Classification Accuracy on Olympic Sports by BDS

| Method | clean&jerk (snatch) | long-jump (triple-jump) | snatch (clean&jerk) | triple-jump (long-jump) |
|---|---|---|---|---|
| **BDS** | 85% (9%) | 80% (2%) | 78% (10%) | 62% (14%) |
| **Holistic** | 73% (21%) | 72% (20%) | 65% (27%) | 38% (43%) |

**Table 3:** Mean Average Precision on Olympic Sports Dataset

| Laptev *et al.* [3] ( BoF ) | Niebles *et al.* [4] ( BDS ) | Liu *et al.* [11] ( Attr. / B+A ) | B+A+D |
|---|---|---|---|
| 62.0% ( 67.8% ) | 72.1% (73.2%) | 74.4% (72.9% / 73.3%) | 76.5% |

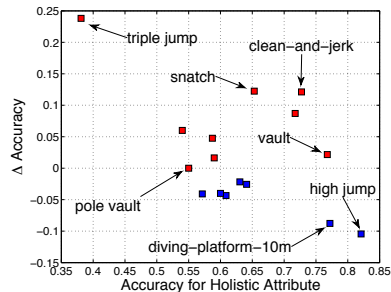

**Figure 4:** Scatter plot of accuracy gain on Olympic Sports by BDS.

of atomic actions, whose temporal structure is critical for discrimination from other classes (*e.g.*, "clean and jerk" *v.s.*"snatch", and "long-jump" *v.s.*"triple-jump"). Since attribute labels are only available for whole sequences, the training sets of the attribute classifiers are much noisier than in the previous experiment. This degrades the quality of attribute models. The dataset was split into 5 subsets, of roughly the same size, and results reported by 5-fold cross-validation. The DTM and TOT classifiers were as above, and all others were implemented with an SVM of kernel $K_\alpha(i,j) = \exp(-\frac{1}{\alpha}d^2(i,j))$, based on the distance measures $d(i,j)$ of the previous section. Table 1 shows that dynamic modeling again has the best performance. However, the gains over the holistic attribute classifier are smaller than in Weizmann. This is due to two factors. First, the noisy attributes make the dynamics harder to model. Note that the robustness of the dynamic models to this noise varies substantially. As before, topic models have the weakest performance and the BDS outperforms the KDS. Second, since fine grained discrimination is not needed for all categories, attribute dynamics are not always necessary. This is confirmed by Figure 4, which presents a scatter plot of the gain (difference in accuracy) of the BDS classifier over the holistic classifier, as a function of the accuracy of the latter. Each point corresponds to an activity. Note the strongly negative correlation between the two factors: *the largest gains occur for the most difficult classes for the holistic classifier*. Table 2 details these results for the two pairs of classes with most confusable attributes. Numbers outside brackets correspond to ground-truth category, numbers in brackets to the confusing class (percentage of ground-truth examples assigned to it). BDS has dramatically better performance for these classes. Overall, despite the attribute noise and the fact that dynamics are not always required for discrimination, the BDS achieves the best performance on this dataset.

Finally, we compare the BDS classifier to classifiers from the literature. Three approaches, representative of the state-of-the art in classification with the BoF [3], dynamic representations [4], and attributes [11], were selected as benchmarks. These were compared to our implementation of BoF (kernel using only word histograms), attributes (the holistic classifier of Table 1), dynamics (the BDS classifier), and multiple kernel classifiers combining 1) BoF and attributes (B+A), and 2) BoF, attributes, and dynamics (B+A+D). All multiple kernels combinations were achieved by cross-validation. The mean average precisions of all 1-vs-all classifiers are reported in Table 3. The numbers in each column report to directly comparable classifiers, *e.g.*, B+A is directly comparable to [11], which jointly classifies BoF histograms and hollistic attribute vectors with a latent SVM. Note that the BDS classifier outperforms the state-of-the-art in dynamic classifiers (Niebles *et al.* [4]), which accounts for the dynamics of the BoF but not action attributes. This holds despite the fact that our attribute categories (only 40 specified attributes) and classifiers (simple SVMs) are much simpler than the best in the literature [11] , which uses both the data-driven and the 40 specified attributes as ours, plus a latent SVM as the classifier. The use of a stronger attribute detection architecture could potentially further improve these results. Note also that the addition of the BDS kernel to the simple attribute representation (B+A+D) far outperforms the use of the more sophisticated attribute classifier of [11], which does not account for attribute dynamics. This illustrates the benefits of modeling the *dynamics of attributes*. The combination of BoF, attributes, and attribute dynamics achieves the overall best performance on this dataset.

## Acknowledgements

This work was partially supported by the NSF award under Grant CCF-0830535. We also thank Jingen Liu for providing the attribute annotations.

# References

[1] J. K. Aggarwal and M. S. Ryoo, "Human activity analysis: A review," *ACM Computing Surveys*, vol. 43, no. 16, pp. 1–16, 2011.

[2] P. Dollár, V. Rabaud, G. Cottrell, and S. Belongie, "Behavior recognition via sparse spatio-temporal features," *ICCV VS-PETS*, 2005.

[3] I. Laptev, M. Marszałek, C. Schmid, and B. Rozenfeld, "Learning realistic human actions from movies," *CVPR*, 2008.

[4] J. C. Niebles, C.-W. Chen, and L. Fei-Fei, "Modeling temporal structure of decomposable motion segments for activity classification," *ECCV*, 2010.

[5] B. Laxton, J. Lim, and D. Kriegman, "Leveraging temporal, contextual and ordering constraints for recognizing complex activities in video," *CVPR*, 2007.

[6] R. Chaudhry, A. Ravichandran, G. Hager, and R. Vidal, "Histograms of oriented optical flow and binet-cauchy kernels on nonlinear dynamical systems for the recognition of human actions," *CVPR*, 2009.

[7] A. Gaidon, Z. Harchaoui, and C. Schmid, "Actom sequence models for efficient action detection," *CVPR*, 2011.

[8] B. Li, M. Ayazoglu, T. Mao, O. Camps, and M. Sznaier, "Activity recognition using dynamic subspace angles," *CVPR*, 2011.

[9] C. H. Lampert, H. Nickisch, and S. Harmeling, "Learning to detect unseen object classes by between-class attribute transfer," *CVPR*, 2009.

[10] N. Rasiwasia and N. Vasconcelos, "Holistic context models for visual recognition," *IEEE Trans. Pattern Analysis and Machine Intelligence*, vol. 34, no. 5, pp. 902–917, 2012.

[11] J. Liu, B. Kuipers, and S. Savarese, "Recognizing human actions by attributes," *CVPR*, 2011.

[12] A. Fathi and G. Mori, "Action recognition by learning mid-level motion features," *CVPR*, 2008.

[13] N. Rasiwasia and N. Vasconcelos, "Holistic context modeling using semantic co-occurrences," *CVPR*, 2009.

[14] A. I. Schein, L. K. Saul, and L. H. Ungar, "A generalized linear model for principal component analysis of binary data," *AISTATS*, 2003.

[15] G. Doretto, A. Chiuso, Y. N. Wu, and S. Soatto, "Dynamic textures," *Int'l J. Computer Vision*, vol. 51, no. 2, pp. 91–109, 2003.

[16] S. V. N. Vishwanathan, A. J. Smola, and R. Vidal, "Binet-cauchy kernels on dynamical systems and its application to the analysis of dynamic scenes," *Int'l J. Computer Vision*, vol. 73, no. 1, pp. 95–119, 2006.

[17] L. Gorelick, M. Blank, E. Shechtman, M. Irani, and R. Basri, "Actions as space-time shapes," *IEEE Trans. Pattern Analysis and Machine Intelligence*, vol. 29, no. 12, pp. 2247–2253, 2007.

[18] N. İkizler and D. A. Forsyth, "Searching for complex human activities with no visual examples," *Int'l J. Computer Vision*, vol. 80, no. 3, pp. 337–357, 2008.

[19] V. Kellokumpu, G. Zhao, and M. Pietikäinen, "Human activity recognition using a dynamic texture based method," *BMVC*, 2008.

[20] N. Rasiwasia and N. Vasconcelos, "Scene classification with low-dimensional semantic spaces and weak supervision," *CVPR*, 2008.

[21] A. Quattoni, M. Collins, and T. Darrell, "Learning visual representations using images with captions," *CVPR*, 2007.

[22] N. Rasiwasi, P. J. Moreno, and N. Vasconcelos, "Bridging the gap: Query by semantic example," *IEEE Trans. Multimedia*, vol. 9, no. 5, pp. 923–938, 2007.

[23] D. M. Blei and J. D. Lafferty, "Dynamic topic models," *ICML*, 2006.

[24] X. Wang and A. McCallum, "Topics over time: a non-markov continuous-time model of topical trends," *ACM SIGKDD*, 2006.

[25] M. Collins, S. Dasgupta, and R. E. Schapire, "A generalization of principal component analysis to the exponential family," *NIPS*, 2002.

[26] R. H. Shumway and D. S. Stoffer, "An approach to time series smoothing and forecasting using the em algorithm," *Journal of Time Series Analysis*, vol. 3, no. 4, pp. 253–264, 1982.

[27] B. Schölkopf, A. Smola, and K.-R. Müller, "Nonlinear component analysis as a kernel eigenvalue problem," *Neural Computation*, vol. 10, pp. 1299–1319, 1998.

[28] A. B. Chan and N. Vasconcelos, "Classifying video with kernel dynamic textures," *CVPR*, 2007.

[29] C.-C. Chang and C.-J. Lin, "LIBSVM: A library for support vector machines," *ACM Trans. on Intelligent Systems and Technology*, vol. 2, no. 3, pp. 27:1–27:27, 2011.

